# Probabilistic methods for Support Vector Machines

**Peter Sollich**
Department of Mathematics, King's College London
Strand, London WC2R 2LS, U.K. Email: peter.sollich@kcl.ac.uk

## Abstract

I describe a framework for interpreting Support Vector Machines (SVMs) as maximum a posteriori (MAP) solutions to inference problems with Gaussian Process priors. This can provide intuitive guidelines for choosing a 'good' SVM kernel. It can also assign (by evidence maximization) optimal values to parameters such as the noise level $C$ which cannot be determined unambiguously from properties of the MAP solution alone (such as cross-validation error). I illustrate this using a simple approximate expression for the SVM evidence. Once $C$ has been determined, error bars on SVM predictions can also be obtained.

## 1 Support Vector Machines: A probabilistic framework

Support Vector Machines (SVMs) have recently been the subject of intense research activity within the neural networks community; for tutorial introductions and overviews of recent developments see [1, 2, 3]. One of the open questions that remains is how to set the 'tunable' parameters of an SVM algorithm: While methods for choosing the width of the kernel function and the noise parameter $C$ (which controls how closely the training data are fitted) have been proposed [4, 5] (see also, very recently, [6]), the effect of the overall shape of the kernel function remains imperfectly understood [1]. Error bars (class probabilities) for SVM predictions — important for safety-critical applications, for example — are also difficult to obtain. In this paper I suggest that a probabilistic interpretation of SVMs could be used to tackle these problems. It shows that the SVM kernel defines a prior over functions on the input space, avoiding the need to think in terms of high-dimensional feature spaces. It also allows one to define quantities such as the evidence (likelihood) for a set of hyperparameters ($C$, kernel amplitude $K_0$ etc). I give a simple approximation to the evidence which can then be maximized to set such hyperparameters. The evidence is sensitive to the values of $C$ and $K_0$ individually, in contrast to properties (such as cross-validation error) of the deterministic solution, which only depends on the product $CK_0$. It can therefore be used to assign an unambiguous value to $C$, from which error bars can be derived.

I focus on two-class classification problems. Suppose we are given a set $D$ of $n$ training examples $(x_i, y_i)$ with binary outputs $y_i = \pm 1$ corresponding to the two classes. The basic SVM idea is to map the inputs $x$ onto vectors $\phi(x)$ in some high-dimensional feature space; ideally, in this feature space, the problem should be linearly separable. Suppose first that this is true. Among all decision hyperplanes $\mathbf{w} \cdot \phi(x) + b = 0$ which separate the training examples (i.e. which obey $y_i(\mathbf{w} \cdot \phi(x_i) + b) > 0$ for all $x_i \in D_X$, $D_X$ being the set of training inputs), the SVM solution is chosen as the one with the largest *margin*, i.e. the largest minimal distance from any of the training examples. Equivalently, one specifies the margin to be one and minimizes the squared length of the weight vector $||\mathbf{w}||^2$ [1], subject to the constraint that $y_i(\mathbf{w} \cdot \phi(x_i) + b) \geq 1$ for all $i$. If the problem is not linearly separable, 'slack variables' $\xi_i \geq 0$ are introduced which measure how much the margin constraints are violated; one writes $y_i(\mathbf{w} \cdot \phi(x_i) + b) \geq 1 - \xi_i$. To control the amount of slack allowed, a penalty term $C \sum_i \xi_i$ is then added to the objective function $\frac{1}{2}||\mathbf{w}||^2$, with a penalty coefficient $C$. Training examples with $y_i(\mathbf{w} \cdot \phi(x_i) + b) \geq 1$ (and hence $\xi_i = 0$) incur no penalty; all others contribute $C[1 - y_i(\mathbf{w} \cdot \phi(x_i) + b)]$ each. This gives the SVM optimization problem: Find $\mathbf{w}$ and $b$ to minimize

$$\frac{1}{2}||\mathbf{w}||^2 + C \sum_i l(y_i[\mathbf{w} \cdot \phi(x_i) + b]) \tag{1}$$

where $l(z)$ is the (shifted) 'hinge loss', $l(z) = (1 - z)\Theta(1 - z)$.

To interpret SVMs probabilistically, one can regard (1) as defining a (negative) log-posterior probability for the parameters $\mathbf{w}$ and $b$ of the SVM, given a training set $D$. The first term gives the prior $Q(\mathbf{w}, b) \sim \exp(-\frac{1}{2}||\mathbf{w}||^2 - \frac{1}{2}b^2 B^{-2})$. This is a Gaussian prior on $\mathbf{w}$; the components of $\mathbf{w}$ are uncorrelated with each other and have unit variance. I have chosen a Gaussian prior on $b$ with variance $B^2$; the flat prior implied by (1) can be recovered[1] by letting $B \to \infty$. Because only the 'latent variable' values $\theta(x) = \mathbf{w} \cdot \phi(x) + b$ — rather than $\mathbf{w}$ and $b$ individually — appear in the second, data dependent term of (1), it makes sense to express the prior directly as a distribution over these. The $\theta(x)$ have a joint Gaussian distribution because the components of $\mathbf{w}$ do, with covariances given by $\langle \theta(x)\theta(x') \rangle = \langle (\phi(x) \cdot \mathbf{w})(\mathbf{w} \cdot \phi(x')) \rangle + B^2 = \phi(x) \cdot \phi(x') + B^2$. The SVM prior is therefore simply a *Gaussian process* (GP) over the functions $\theta$, with covariance function $K(x, x') = \phi(x) \cdot \phi(x') + B^2$ (and zero mean). This correspondence between SVMs and GPs has been noted by a number of authors, *e.g.* [6, 7, 8, 9, 10].

The second term in (1) becomes a (negative) log-likelihood if we define the probability of obtaining output $y$ for a given $x$ (and $\theta$) as

$$Q(y = \pm 1 | x, \theta) = \kappa(C) \exp[-Cl(y\theta(x))] \tag{2}$$

We set $\kappa(C) = 1/[1 + \exp(-2C)]$ to ensure that the probabilities for $y = \pm 1$ never add up to a value larger than one. The likelihood for the complete data set is then $Q(D|\theta) = \prod_i Q(y_i|x_i, \theta)Q(x_i)$, with some input distribution $Q(x)$ which remains essentially arbitrary at this point. However, this likelihood function is not normalized, because

$$\nu(\theta(x)) = Q(1|x, \theta) + Q(-1|x, \theta) = \kappa(C)\{\exp[-Cl(\theta(x))] + \exp[-Cl(-\theta(x))]\} < 1$$

except when $|\theta(x)| = 1$. To remedy this, I write the actual probability model as

$$P(D, \theta) = Q(D|\theta)Q(\theta)/\mathcal{N}(D). \tag{3}$$

Its posterior probability $P(\theta|D) \sim Q(D|\theta)Q(\theta)$ is independent of the normalization factor $\mathcal{N}(D)$; by construction, the MAP value of $\theta$ is therefore the SVM solution. The simplest choice of $\mathcal{N}(D)$ which normalizes $P(D, \theta)$ is $D$-independent:

$$\mathcal{N} = \overline{N^n} = \int d\theta \, Q(\theta)N^n(\theta), \quad N(\theta) = \int dx \, Q(x) \, \nu(\theta(x)). \tag{4}$$

Conceptually, this corresponds to the following procedure of sampling from $P(D, \theta)$: First, sample $\theta$ from the GP prior $Q(\theta)$. Then, for each data point, sample $x$ from $Q(x)$. Assign outputs $y = \pm 1$ with probability $Q(y|x, \theta)$, respectively; with the *remaining* probability $1 - \nu(\theta(x))$ (the 'don't know' class probability in [11]), restart the whole process by sampling a new $\theta$. Because $\nu(\theta(x))$ is smallest[2] inside the 'gap' $|\theta(x)| < 1$, functions $\theta$ with many values in this gap are less likely to 'survive' until a dataset of the required size $n$ is built up. This is reflected in an $n$-dependent factor in the (effective) prior, which follows from (3,4) as $P(\theta) \sim Q(\theta)N^n(\theta)$. Correspondingly, in the likelihood

$$P(y|x, \theta) = Q(y|x, \theta)/\nu(\theta(x)), \qquad P(x|\theta) \sim Q(x) \, \nu(\theta(x)) \tag{5}$$

(which now is normalized over $y = \pm 1$), the input density is influenced by the function $\theta$ itself; it is reduced in the 'uncertainty gaps' $|\theta(x)| < 1$.

To summarize, eqs. (2-5) define a probabilistic data generation model whose MAP solution $\theta^* = \text{argmax} \, P(\theta|D)$ for a given data set $D$ is identical to a standard SVM. The effective prior $P(\theta)$ is a GP prior modified by a data set size-dependent factor; the likelihood (5) defines not just a conditional output distribution, but also an input distribution (relative to some arbitrary $Q(x)$). All relevant properties of the feature space are encoded in the underlying GP prior $Q(\theta)$, with covariance matrix equal to the kernel $K(x, x')$. The log-posterior of the model

$$\ln P(\theta|D) = -\tfrac{1}{2} \int dx \, dx' \, \theta(x)K^{-1}(x, x') \, \theta(x') - C \sum_i l(y_i\theta(x_i)) + \text{const} \tag{6}$$

is just a transformation of (1) from $\mathbf{w}$ and $b$ to $\theta$. By differentiating w.r.t. the $\theta(x)$ for non-training inputs, one sees that its maximum is of the standard form $\theta^*(x) = \sum_i \alpha_i y_i K(x, x_i)$; for $y_i\theta^*(x_i) > 1$, $< 1$, and $= 1$ one has $\alpha_i = 0$, $\alpha_i = C$ and $\alpha_i \in [0, C]$ respectively. I will call the training inputs $x_i$ in the last group marginal; they form a subset of all support vectors (the $x_i$ with $\alpha_i > 0$). The sparseness of the SVM solution (often the number of support vectors is $\ll n$) comes from the fact that the hinge loss $l(z)$ is constant for $z > 1$. This contrasts with other uses of GP models for classification (see *e.g.* [12]), where instead of the likelihood (2) a sigmoidal (often logistic) 'transfer function' with nonzero gradient everywhere is used. Moreover, in the noise free limit, the sigmoidal transfer function becomes a step function, and the MAP values $\theta^*$ will tend to the trivial solution $\theta^*(x) = 0$. This illuminates from an alternative point of view why the margin (the 'shift' in the hinge loss) is important for SVMs.

Within the probabilistic framework, the main effect of the kernel in SVM classification is to change the properties of the underlying GP prior $Q(\theta)$ in $P(\theta) \sim$

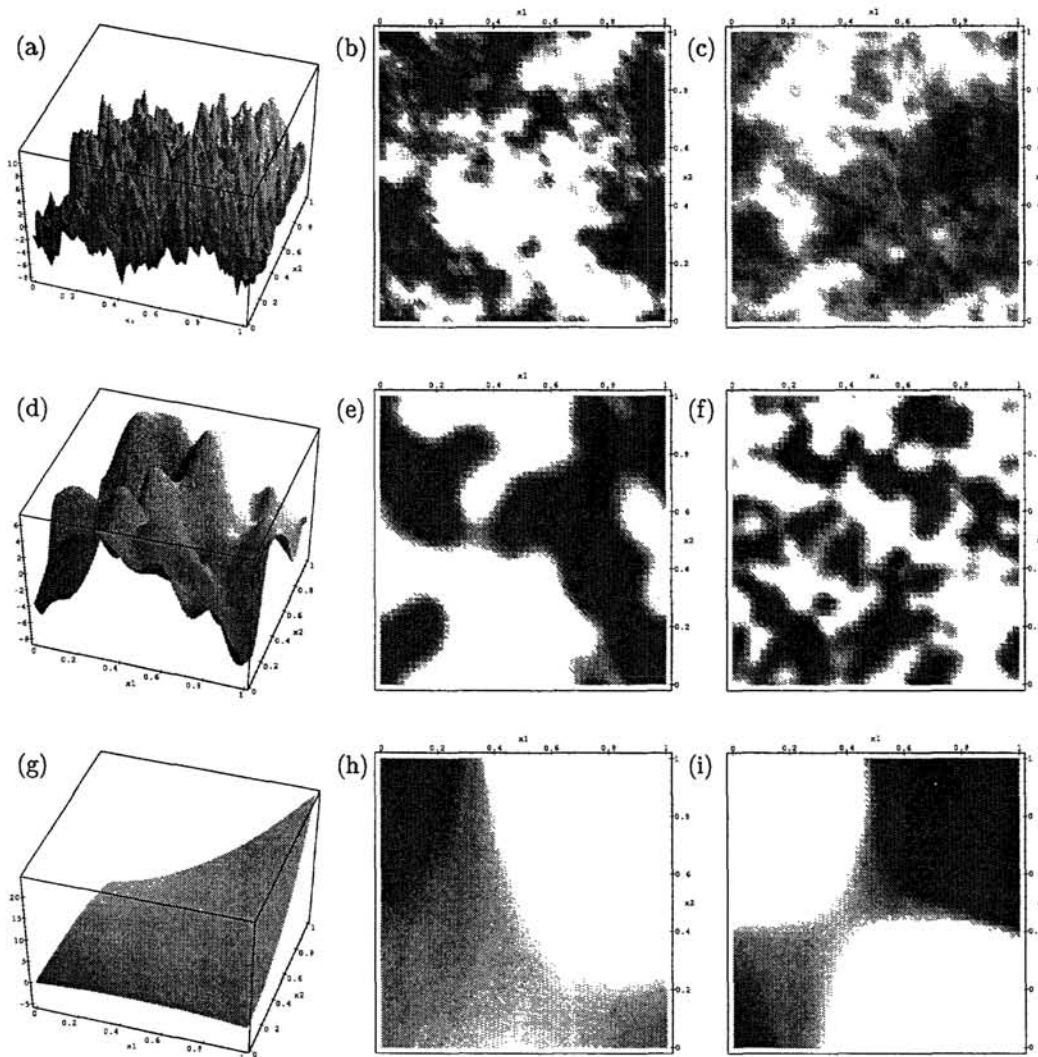

Figure 1: Samples from SVM priors; the input space is the unit square $[0,1]^2$. 3d plots are samples $\theta(x)$ from the underlying Gaussian process prior $Q(\theta)$. 2d greyscale plots represent the output distributions obtained when $\theta(x)$ is used in the likelihood model (5) with $C = 2$; the greyscale indicates the probability of $y = 1$ (black: 0, white: 1). (a,b) Exponential (Ornstein-Uhlenbeck) kernel/covariance function $K_0 \exp(-|x - x'|/l)$, giving rough $\theta(x)$ and decision boundaries. Length scale $l = 0.1$, $K_0 = 10$. (c) Same with $K_0 = 1$, i.e. with a reduced amplitude of $\theta(x)$; note how, in a sample from the prior corresponding to this new kernel, the grey 'uncertainty gaps' (given roughly by $|\theta(x)| < 1$) between regions of definite outputs (black/white) have widened. (d,e) As first row, but with squared exponential (RBF) kernel $K_0 \exp[-(x - x')^2/(2l^2)]$, yielding smooth $\theta(x)$ and decision boundaries. (f) Changing $l$ to 0.05 (while holding $K_0$ fixed at 10) and taking a new sample shows how this parameter sets the typical length scale for decision regions. (g,h) Polynomial kernel $(1 + x \cdot x')^p$, with $p = 5$; (i) $p = 10$. The absence of a clear length scale and the widely differing magnitudes of $\theta(x)$ in the bottom left ($x = [0, 0]$) and top right ($x = [1, 1]$) corners of the square make this kernel less plausible from a probabilistic point of view.

$Q(\theta)N^n(\theta)$. Fig. 1 illustrates this with samples from $Q(\theta)$ for three different types of kernels. The effect of the kernel on smoothness of decision boundaries, and typical sizes of decision regions and 'uncertainty gaps' between them, can clearly be seen. When prior knowledge about these properties of the target is available, the probabilistic framework can therefore provide intuition for a suitable choice of kernel. Note that the samples in Fig. 1 are from $Q(\theta)$, rather than from the effective prior $P(\theta)$. One finds, however, that the $n$-dependent factor $N^n(\theta)$ does not change the properties of the prior qualitatively[3].

## 2 Evidence and error bars

Beyond providing intuition about SVM kernels, the probabilistic framework discussed above also makes it possible to apply Bayesian methods to SVMs. For example, one can define the evidence, i.e. the likelihood of the data $D$, given the model as specified by the hyperparameters $C$ and (some parameters defining) $K(x, x')$. It follows from (3) as

$$P(D) = Q(D)/\overline{N^n}, \qquad Q(D) = \int d\theta \, Q(D|\theta)Q(\theta). \tag{7}$$

The factor $Q(D)$ is the 'naive' evidence derived from the unnormalized likelihood model; the correction factor $\overline{N^n}$ ensures that $P(D)$ is normalized over all data sets. This is crucial in order to guarantee that optimization of the (log) evidence gives optimal hyperparameter values at least on average (M Opper, private communication). Clearly, $P(D)$ will in general depend on $C$ and $K(x, x')$ separately. The actual SVM solution, on the other hand, i.e. the MAP values $\theta^*$, can be seen from (6) to depend on the product $CK(x, x')$ only. Properties of the deterministically trained SVM alone (such as test or cross-validation error) cannot therefore be used to determine $C$ and the resulting class probabilities (5) unambiguously.

I now outline how a simple approximation to the naive evidence can be derived. $Q(D)$ is given by an integral over all $\theta(x)$, with the log integrand being (6) up to an additive constant. After integrating out the Gaussian distributed $\theta(x)$ with $x \notin D_X$, an intractable integral over the $\theta(x_i)$ remains. However, progress can be made by expanding the log integrand around its maximum $\theta^*(x_i)$. For all non-marginal training inputs this is equivalent to Laplace's approximation: the first terms in the expansion are quadratic in the deviations from the maximum and give simple Gaussian integrals. For the remaining $\theta(x_i)$, the leading terms in the log integrand vary *linearly* near the maximum. Couplings between these $\theta(x_i)$ only appear at the next (quadratic) order; discarding these terms as subleading, the integral factorizes over the $\theta(x_i)$ and can be evaluated. The end result of this calculation is:

$$\ln Q(D) \approx -\tfrac{1}{2} \sum_i y_i \alpha_i \theta^*(x_i) - C \sum_i l(y_i \theta^*(x_i)) - n \ln(1 + e^{-2C}) - \tfrac{1}{2} \ln \det(\mathbf{L}_m \mathbf{K}_m) \tag{8}$$

The first three terms represent the maximum of the log integrand, $\ln Q(D|\theta^*)$; the last one comes from the integration over the fluctuations of the $\theta(x)$. Note that it only contains information about the *marginal* training inputs: $\mathbf{K}_m$ is the corresponding submatrix of $K(x, x')$, and $\mathbf{L}_m$ is a diagonal matrix with entries

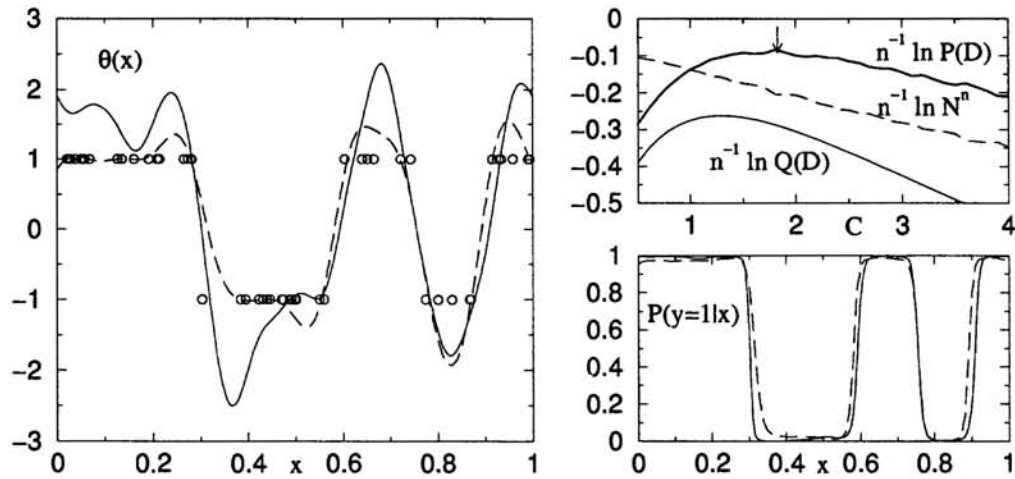

Figure 2: Toy example of evidence maximization. Left: Target 'latent' function $\theta(x)$ (solid line). A SVM with RBF kernel $K(x,x') = K_0 \exp[-(x-x')^2/(2l^2)]$, $l = 0.05$, $CK_0 = 2.5$ was trained (dashed line) on $n = 50$ training examples (circles). Keeping $CK_0$ constant, the evidence $P(D)$ (top right) was then evaluated as a function of $C$ using (7,8). Note how the normalization factor $\overline{N^n}$ shifts the maximum of $P(D)$ towards larger values of $C$ than in the naive evidence $Q(D)$. Bottom right: Class probability $P(y=1|x)$ for the target (solid), and prediction at the evidence maximum $C \approx 1.8$ (dashed). The target was generated from (3) with $C=2$.

$2\pi[\alpha_i(C-\alpha_i)/C]^2$. Given the sparseness of the SVM solution, these matrices should be reasonably small, making their determinants amenable to numerical computation or estimation [12]. Eq. (8) diverges when $\alpha_i \to 0$ or $\to C$ for one of the marginal training inputs; the approximation of retaining only linear terms in the log integrand then breaks down. I therefore adopt the simple heuristic of replacing $\det(\mathbf{L}_m\mathbf{K}_m)$ by $\det(\mathbf{I} + \mathbf{L}_m\mathbf{K}_m)$, which prevents these spurious singularities ($\mathbf{I}$ is the identity matrix). This choice also keeps the evidence continuous when training inputs move in or out of the set of marginal inputs as hyperparameters are varied.

Fig. 2 shows a simple application of the evidence estimate (8). For a given data set, the evidence $P(D)$ was evaluated[4] as a function of $C$. The kernel amplitude $K_0$ was varied simultaneously such that $CK_0$ and hence the SVM solution itself remained unchanged. Because the data set was generated artificially from the probability model (3), the 'true' value of $C = 2$ was known; in spite of the rather crude approximation for $Q(D)$, the maximum of the full evidence $P(D)$ identifies $C \approx 1.8$ quite close to the truth. The approximate class probability prediction $P(y = 1|x, D)$ for this value of $C$ is also plotted in Fig. 2; it overestimates the noise in the target somewhat. Note that $P(y|x, D)$ was obtained simply by inserting the MAP values $\theta^*(x)$ into (5). In a proper Bayesian treatment, an average over the posterior distribution $P(\theta|D)$ should of course be taken; I leave this for future work.

[4]The normalization factor $\overline{N^n}$ was estimated, for the assumed uniform input density $Q(x)$ of the example, by sampling from the GP prior $Q(\theta)$. If $Q(x)$ is unknown, the empirical training input distribution can be used as a proxy, and one samples instead from a multivariate Gaussian for the $\theta(x_i)$ with covariance matrix $K(x_i, x_j)$. This gave very similar values of $\ln \overline{N^n}$ in the example, even when only a subset of 30 training inputs was used.

In summary, I have described a probabilistic framework for SVM classification. It gives an intuitive understanding of the effect of the kernel, which determines a Gaussian process prior. More importantly, it also allows a properly normalized evidence to be defined; from this, optimal values of hyperparameters such as the noise parameter $C$, and corresponding error bars, can be derived. Future work will have to include more comprehensive experimental tests of the simple Laplace-type estimate of the (naive) evidence $Q(D)$ that I have given, and comparison with other approaches. These include variational methods; very recent experiments with a Gaussian approximation for the posterior $P(\theta|D)$, for example, seem promising [6]. Further improvement should be possible by dropping the restriction to a 'factor-analysed' covariance form [6]. (One easily shows that the optimal Gaussian covariance matrix is $(\mathbf{D} + \mathbf{K}^{-1})^{-1}$, parameterized only by a diagonal matrix $\mathbf{D}$.) It will also be interesting to compare the Laplace and Gaussian variational results for the evidence with those from the 'cavity field' approach of [10].

## Acknowledgements

It is a pleasure to thank Tommi Jaakkola, Manfred Opper, Matthias Seeger, Chris Williams and Ole Winther for interesting comments and discussions, and the Royal Society for financial support through a Dorothy Hodgkin Research Fellowship.

## Footnotes

[1]In the probabilistic setting, it actually makes more sense to keep $B$ finite (and small); for $B \to \infty$, only training sets with all $y_i$ equal have nonzero probability.

[2]This is true for $C > \ln 2$. For smaller $C$, $\nu(\theta(x))$ is actually higher in the gap, and the model makes less intuitive sense.

[3]Quantitative changes arise because function values with $|\theta(x)| < 1$ are 'discouraged' for large $n$; this tends to increase the size of the decision regions and narrow the uncertainty gaps. I have verified this by comparing samples from $Q(\theta)$ and $P(\theta)$.

# References

[1] C J C Burges. A tutorial on support vector machines for pattern recognition. *Data Mining and Knowledge Discovery*, 2:121–167, 1998.

[2] A J Smola and B Schölkopf. A tutorial on support vector regression. 1998. Neuro COLT Technical Report TR-1998-030; available from http://svm.first.gmd.de/.

[3] B Schölkopf, C Burges, and A J Smola. *Advances in Kernel Methods: Support Vector Machines*. MIT Press, Cambridge, MA, 1998.

[4] B Schölkopf, P Bartlett, A Smola, and R Williamson. Shrinking the tube: a new support vector regression algorithm. In *NIPS 11*.

[5] N Cristianini, C Campbell, and J Shawe-Taylor. Dynamically adapting kernels in support vector machines. In *NIPS 11*.

[6] M Seeger. Bayesian model selection for Support Vector machines, Gaussian processes and other kernel classifiers. Submitted to *NIPS 12*.

[7] G Wahba. Support vector machines, reproducing kernel Hilbert spaces and the randomized GACV. Technical Report 984, University of Wisconsin, 1997.

[8] T S Jaakkola and D Haussler. Probabilistic kernel regression models. In *Proceedings of The 7th International Workshop on Artificial Intelligence and Statistics*. To appear.

[9] A J Smola, B Schölkopf, and K R Müller. The connection between regularization operators and support vector kernels. *Neural Networks*, 11:637–649, 1998.

[10] M Opper and O Winther. Gaussian process classification and SVM: Mean field results and leave-one-out estimator. In *Advances in Large Margin Classifiers*. MIT Press. To appear.

[11] P Sollich. Probabilistic interpretation and Bayesian methods for Support Vector Machines. Submitted to ICANN 99.

[12] C K I Williams. Prediction with Gaussian processes: From linear regression to linear prediction and beyond. In M I Jordan, editor, *Learning and Inference in Graphical Models*, pages 599–621. Kluwer Academic, 1998.